# A CONNECTIONIST EXPERT SYSTEM
# THAT ACTUALLY WORKS

Gary Bradshaw
Psychology

Richard Fozzard
Computer Science
University of Colorado
Boulder, CO 80302
fozzard@boulder.colorado.edu

Louis Ceci
Computer Science

## ABSTRACT

The Space Environment Laboratory in Boulder has collaborated with the University of Colorado to construct a small expert system for solar flare forecasting, called THEO. It performed as well as a skilled human forecaster. We have constructed *TheoNet*, a three-layer back-propagation connectionist network that learns to forecast flares as well as THEO does. *TheoNet*'s success suggests that a connectionist network can perform the task of knowledge engineering automatically. A study of the internal representations constructed by the network may give insights to the "microstructure" of reasoning processes in the human brain.

## INTRODUCTION

Can neural network learning algorithms let us build "expert systems" automatically, merely by presenting the network with data from the problem domain? We tested this possibility in a domain where a traditional expert system has been developed that is at least as good as the expert, to see if the connectionist approach could stand up to tough competition.

Knowledge-based expert systems attempt to capture in a computer program the knowledge of a human expert in a limited domain and make this knowledge available to a user with less experience. Such systems could be valuable as an assistant to a forecaster or as a training tool. In the past three years, the Space Environment Laboratory (SEL) in Boulder has collaborated with the Computer Science and Psychology Departments at the University of Colorado to construct a small expert system emulating a methodology for solar flare forecasting developed by Pat McIntosh, senior solar physicist at SEL. The project convincingly demonstrated the possibilities of this type of computer assistance, which also proved to be a useful tool for formally expressing a methodology, verifying its performance, and instructing novice forecasters. The system,

named THEO (an OPS-83 production system with about 700 rules), performed as well as a skilled human forecaster using the same methods, and scored well compared with actual forecasts in the period covered by the test data [Lewis and Dennett 1986].

In recent years connectionist (sometimes called "non-symbolic" or "neural") network approaches have been used with varying degrees of success to simulate human behavior in such areas as vision and speech learning and recognition [Hinton 1987, Lehky and Sejnowski 1988, Sejnowski and Rosenberg 1986, Elman and Zipser 1987]. Logic (or "symbolic") approaches have been used to simulate human (especially expert) reasoning [see Newell 1980 and Davis 1982]. There has developed in the artificial intelligence and cognitive psychology communities quite a schism between the two areas of research and the same problem has rarely been attacked by both approaches. It is hardly our intent to debate the relative merits of the two paradigms. The intent of this project is to directly apply a connectionist learning technique (multi-layer back-propagation) to the same problem, even the very same database used in an existing successful rule-based expert system. At this time we know of no current work attempting to do this.

Forecasting, as described by those who practice it, is a unique combination of informal reasoning within very soft constraints supplied by often incomplete and inaccurate data. The type of reasoning involved makes it a natural application for traditional rule-based approaches. Solar and flare occurrence data are often inconsistent and noisy. The nature of the data, therefore, calls for careful handling of rule strengths and certainty factors. Yet dealing with this sort of data is exactly one of the strengths claimed for connectionist networks. It may also be that some of the reasoning involves pattern matching of the different categories of data. This is what led us to hope that a connectionist network might be able to learn the necessary internal representations to cope with this task.

## TECHNICAL APPROACH

The *TheoNet* network model has three layers of simple, neuron-like processing elements called "units". The lowest layer is the input layer and is clamped to a pattern that is a distributed representation of the solar data for a given day. For the middle ("hidden") and upper ("output") layers, each unit's output (called "activation") is the weighted sum of all inputs from the units in the layer below:

$$y_j = \frac{1}{1 + e^{-x_j}} \quad \text{where:} \quad x_j = \sum_i y_i w_{ji} - \theta_j \quad (1)$$

where $y_i$ is the activation of the $i$th unit in the layer below, $w_{ji}$ is the weight on the connection from the $i$th to the $j$th unit, and $\theta_j$ is the threshold of the $j$th

unit. The weights are initially set to random values between -1.0 and +1.0, but are allowed to vary beyond that range. A least mean square error learning procedure called *back-propagation* is used to modify the weights incrementally for each input data pattern presented to the network. This compares the output unit activations with the "correct" (what actually happened) solar flare activity for that day. This gives the weight update rule:

$$\Delta w_{ji}(t+1) = -\varepsilon \nabla E(t) + \alpha \Delta w_{ji}(t) \tag{2}$$

where $\nabla E(t)$ is the partial derivative of least mean square error, $\varepsilon$ is a parameter called the *learning rate* that affects how quickly the network attempts to converge on the appropriate weights (if possible), and $\alpha$ is called the *momentum* which affects the amount of damping in the procedure. This is as in [Hinton 1987], except that no weight decay was used. Weights were updated after each presentation of an input/output pattern.

The network was constructed as shown in Figure 1. The top three output units are intended to code for each of the three classes of solar flares to be forecasted. The individual activations are currently intended to correspond to the relative likelihood of a flare of that class within the next 24 hours (see the analysis of the results below). The 17 input units provide a distributed coding of the ten categories of input data that are currently fed into the "default" mode of the expert system THEO. That is, three binary (on/off) units code for the seven classes of sunspots, two for spot distribution, and so on. The hidden units mediate the transfer of activation from the input to the output units and provide the network with the potential of forming internal representations. Each layer is fully interconnected with the layer above and/or below it, but there are no connections within layers.

## RESULTS

The P3 connectionist network simulator from David Zipser of University of California at San Diego's parallel distributed processing (PDP) group was used to implement and test *TheoNet* on a Symbolics 3653 workstation. This simulator allowed the use of Lisp code to compile statistics and provided an interactive environment for working with the network simulation.

The network was trained and tested using two sets of data of about 500 input/output pairs (solar data/flare occurrence) each from the THEO database. Many of these resulted in the same input pattern (there were only about 250 different input patterns total), and in many cases the same input would result in different flare results in the following 24 hours. The data was from a low flare frequency period (about 70-80 flares total). These sorts of inconsistencies in the data make the job of prediction difficult to systematize. The network would be

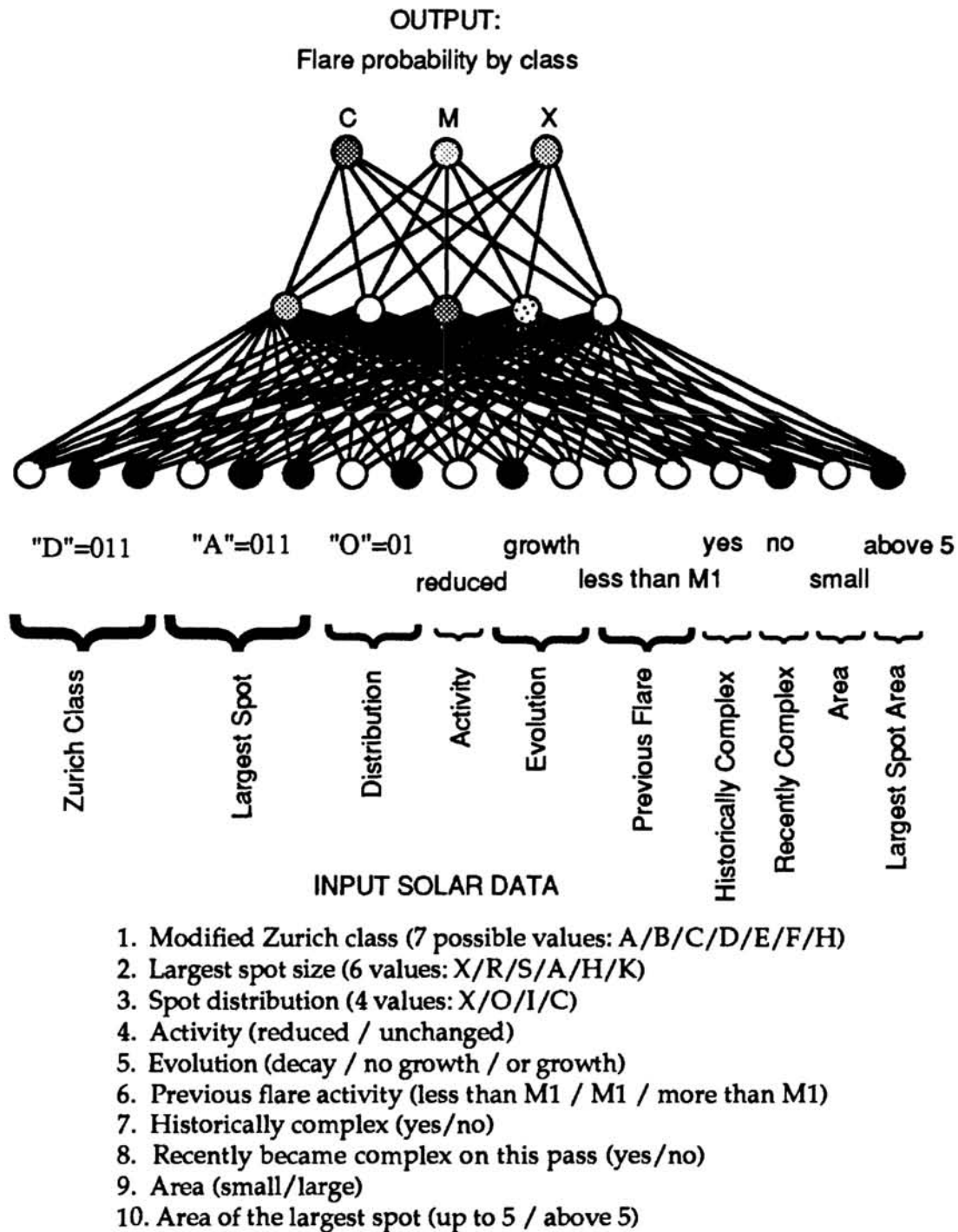

1. Modified Zurich class (7 possible values: A/B/C/D/E/F/H)
2. Largest spot size (6 values: X/R/S/A/H/K)
3. Spot distribution (4 values: X/O/I/C)
4. Activity (reduced / unchanged)
5. Evolution (decay / no growth / or growth)
6. Previous flare activity (less than M1 / M1 / more than M1)
7. Historically complex (yes/no)
8. Recently became complex on this pass (yes/no)
9. Area (small/large)
10. Area of the largest spot (up to 5 / above 5)

**Figure 1.** Architecture of *TheoNet*

trained on one data set and then tested on the other (it did not matter which one was used for which).

Two ways of measuring performance were used. An earlier simulation tracked a simple measure called *overall-prediction-error*. This was the average difference over one complete epoch of input patterns between the activation of an output unit and the "correct" value it was supposed to have. This is directly related to the sum-squared error used by the back-propagation method.

While the overall-prediction-error would drop quickly for all flare classes after a dozen epoches or so (about 5 minutes on the Symbolics), individual weights would take much longer to stabilize. Oscillations were seen in weight values if a large learning rate was used. When this was reduced to 0.2 or lower (with a momentum of 0.9), the weights would converge more smoothly to their final values.

Overall-prediction-error however, is not a good measure of performance since this could be reduced simply by reducing average activation (a "Just-Say-No" network). Analyzing performance of an expert system is best done using measures from the problem domain. Forecasting problems are essentially probabilistic, requiring the detection of signal from noisy data. Thus forecasting techniques and systems are often analyzed using signal detection theory [Spoehr and Lehmkuhle 1982].

The system was modified to calculate $P(H)$, the probability of a hit, and $P(FA)$, the probability of a false alarm, over each epoch. These parameters depend on the *response bias*, which determines the activation level used as a threshold for a yes/no response[*]. A graph of $P(H)$ versus $P(FA)$ gives the *receiver operating characteristic* or ROC curve. The amount that this curve is bowed away from a 1:1 slope is the degree to which a signal is being detected against background. This was the method used for measuring the performance of THEO [Lewis and Dennett 1986].

As in the earlier simulation, the network was exposed to the test data before and after training. After training, the probability of hits was consistently higher than that of false alarms in all flare classes (Figure 2). Given the limited data and very low activations for X-class flares, it may or may not be reasonable to draw conclusions about the network's ability to detect these - in the test data set there were only four X-flares in the entire data set. The degree to which the hits exceed false alarms is given by $a'$, the area under the curve. The performance of *TheoNet* was at least as good as the THEO expert system.

---

[*] Even though both THEO and *TheoNet* have a continuous output (probability of flare and activation), varying the response bias gives a continuous evaluation of performance at any output level.

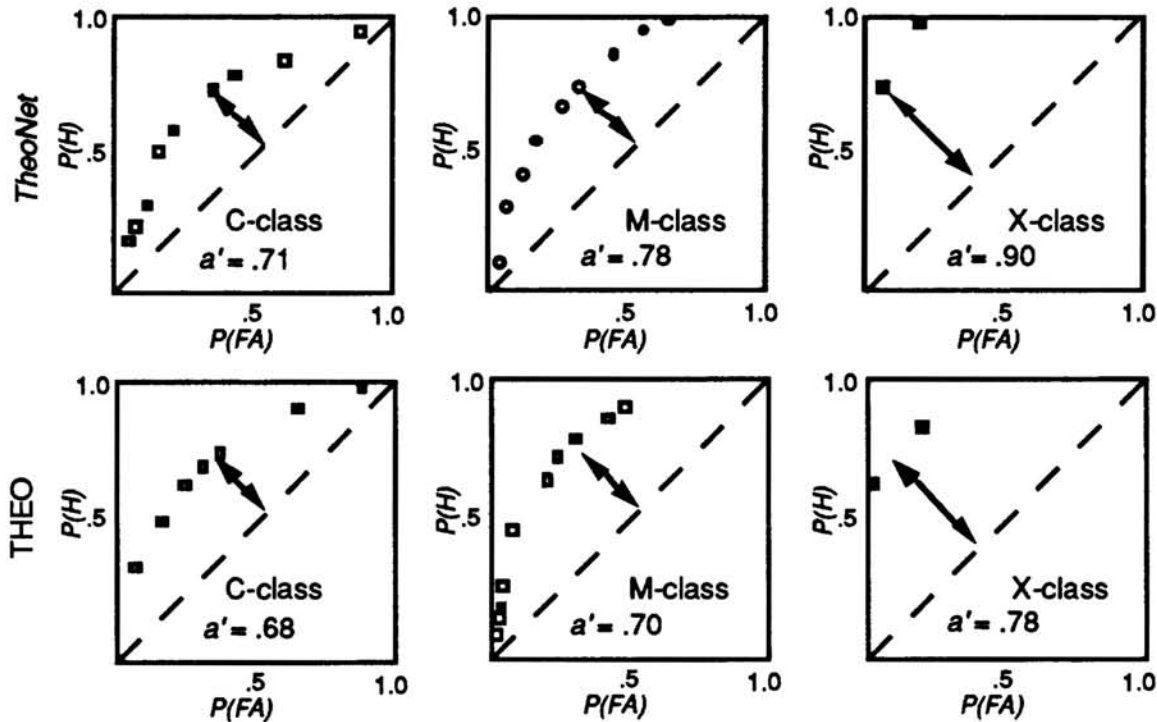

**Figure 2.** ROC performance measures of *TheoNet* and THEO

## CONCLUSIONS

Two particularly intriguing prospects are raised by these results. The first is that if a connectionist network can perform the same task as a rule-based system, then a study of the internal representations constructed by the network may give insights to the "microstructure" of how reasoning processes occur in the human brain. These are the same reasoning processes delineated at a higher level of description by the rules in an expert system. How this sort of work might affect the schism between the symbolic and non-symbolic camps (mentioned in the introduction) is anyone's guess. Our hope is that the two paradigms may eventually come to complement and support each other in cognitive science research.

The second prospect is more of an engineering nature. Though connectionist networks do offer some amount of biological plausibility (and hence their trendy status right now), it is difficult to imagine a neural mechanism for the back-propagation algorithm. However, what do engineers care? As a lot, they are more interested in implementing a solution than explaining the nature of human thought. Witness the current explosion of expert system technology in the marketplace today. Yet for all its glamor, expert systems have usually proved time consuming and expensive to implement. The "knowledge-engineering" step of interviewing experts and transferring their knowledge to

rules that work successfully together has been the most difficult and expensive part, even with advanced knowledge representation languages and expert system shells. *TheoNet* has shown that at least in this instance, a standard back-propagation network can quickly learn those necessary representations and interactions (rules?) needed to do the same sort of reasoning. Development of THEO (originally presented as one of the quickest developments of a usable expert system) required more than a man-year of work and 700 rules, while *TheoNet* was developed in less than a week using a simple simulator. In addition, THEO requires about five minutes to process a single prediction while the network requires only a few milliseconds, thus promising better performance under real-time conditions.

Many questions remain to be answered. *TheoNet* has only been tested on a small segment of the 11-year solar cycle. It has yet to be determined how many hidden units are needed for generalization of performance (is a simple pattern associator sufficient?). We would like to examine the internal representations formed and see if there is any relationship to the rules in THEO. Without those interpretations, connectionist networks cannot easily offer the help and explanation facilities of traditional expert systems that are a fallout of the rule-writing process.

Since the categories of data used were what is input to THEO, and therefore known to be significant, we need to ask if the network can eliminate redundant or unnecessary categories. We also would like to attempt to implement other well-known expert systems to determine the generality of this approach.

## Acknowledgements

The authors would like to acknowledge the encouragement and advice of Paul Smolensky (University of Colorado) on this project and the desktop publishing equipment of Fischer Imaging Corporation in Denver.

# REFERENCES

Randall Davis "Expert Systems: Where Are We? And Where Do We Go From Here?", *The AI Magazine*, Spring 1982

J.L. Elman and David Zipser *Discovering the Hidden Structure of Speech* ICS Technical Report 8701, University of California, San Diego

Geoffrey Hinton "Learning Translation Invariant Recognition in a Massively Parallel Network", in *Proc. Conf. Parallel Architectures and Languages Europe*, Eindhoven, The Netherlands, June 1987

Sidney Lehky and Terrence Sejnowski, "Neural Network Model for the Cortical Representation of Surface Curvature from Images of Shaded Surfaces", in *Sensory Processing*, J.S. Lund, ed., Oxford 1988

Clayton Lewis and Joann Dennett "Joint CU/NOAA Study Predicts Events on the Sun with Artificial Intelligence Technology", *CUEngineering*, 1986

Allen Newell "Physical Symbol Systems", *Cognitive Science* 4:135-183

David Rumelhart, Jay McClelland, and the PDP research group, *Parallel Distributed Processing. Volume 1.* Cambridge, MA, Bradford books, 1986

Terrence Sejnowski and C.R. Rosenberg, *NETtalk: A parallel network that learns to read aloud* Technical Report 86-01 Dept. of Electrical Engineering and Computer Science, Johns Hopkins University, Baltimore MD

K.T. Spoehr and S.W. Lehmkuhle "Signal Detection Theory" in *Visual Information Processing*, Freeman 1982
